# Convex Repeated Games and Fenchel Duality

**Shai Shalev-Shwartz**[1] **and Yoram Singer**[1,2]

[1] School of Computer Sci. & Eng., The Hebrew University, Jerusalem 91904, Israel
[2] Google Inc. 1600 Amphitheater Parkway, Mountain View, CA 94043, USA

## Abstract

We describe an algorithmic framework for an abstract game which we term a convex repeated game. We show that various online learning and boosting algorithms can be all derived as special cases of our algorithmic framework. This unified view explains the properties of existing algorithms and also enables us to derive several new interesting algorithms. Our algorithmic framework stems from a connection that we build between the notions of *regret* in game theory and *weak duality* in convex optimization.

## 1  Introduction and Problem Setting

Several problems arising in machine learning can be modeled as a convex repeated game. Convex repeated games are closely related to *online convex programming* (see [19, 9] and the discussion in the last section). A convex repeated game is a two players game that is performed in a sequence of consecutive rounds. On round $t$ of the repeated game, the first player chooses a vector $\mathbf{w}_t$ from a convex set $S$. Next, the second player responds with a convex function $g_t : S \rightarrow \mathbb{R}$. Finally, the first player suffers an instantaneous loss $g_t(\mathbf{w}_t)$. We study the game from the viewpoint of the first player. The goal of the first player is to minimize its cumulative loss, $\sum_t g_t(\mathbf{w}_t)$.

To motivate this rather abstract setting let us first cast the more familiar setting of online learning as a convex repeated game. Online learning is performed in a sequence of consecutive rounds. On round $t$, the learner first receives a question, cast as a vector $\mathbf{x}_t$, and is required to provide an answer for this question. For example, $\mathbf{x}_t$ can be an encoding of an email message and the question is whether the email is spam or not. The prediction of the learner is performed based on an hypothesis, $h_t : \mathcal{X} \rightarrow \mathcal{Y}$, where $\mathcal{X}$ is the set of questions and $Y$ is the set of possible answers. In the aforementioned example, $\mathcal{Y}$ would be $\{+1, -1\}$ where $+1$ stands for a spam email and $-1$ stands for a benign one. After predicting an answer, the learner receives the correct answer for the question, denoted $y_t$, and suffers loss according to a loss function $\ell(h_t, (\mathbf{x}_t, y_t))$. In most cases, the hypotheses used for prediction come from a parameterized set of hypotheses, $H = \{h_{\mathbf{w}} : \mathbf{w} \in S\}$. For example, the set of linear classifiers, which is used for answering yes/no questions, is defined as $H = \{h_{\mathbf{w}}(\mathbf{x}) = \text{sign}(\langle \mathbf{w}, \mathbf{x} \rangle) : \mathbf{w} \in \mathbb{R}^n\}$. Thus, rather than saying that on round $t$ the learner chooses a hypothesis, we can say that the learner chooses a vector $\mathbf{w}_t$ and its hypothesis is $h_{\mathbf{w}_t}$. Next, we note that once the environment chooses a question-answer pair $(\mathbf{x}_t, y_t)$, the loss function becomes a function over the hypotheses space or equivalently over the set of parameter vectors $S$. We can therefore redefine the online learning process as follows. On round $t$, the learner chooses a vector $\mathbf{w}_t \in S$, which defines a hypothesis $h_{\mathbf{w}_t}$ to be used for prediction. Then, the environment chooses a question-answer pair $(\mathbf{x}_t, y_t)$, which induces the following loss function over the set of parameter vectors, $g_t(\mathbf{w}) = \ell(h_{\mathbf{w}}, (\mathbf{x}_t, y_t))$. Finally, the learner suffers the loss $g_t(\mathbf{w}_t) = \ell(h_{\mathbf{w}_t}, (\mathbf{x}_t, y_t))$. We have therefore described the process of online learning as a convex repeated game.

In this paper we assess the performance of the first player using the notion of *regret*. Given a number of rounds $T$ and a fixed vector $\mathbf{u} \in S$, we define the regret of the first player as the excess loss for

not consistently playing the vector $\mathbf{u}$,

$$\frac{1}{T}\sum_{t=1}^{T} g_t(\mathbf{w}_t) - \frac{1}{T}\sum_{t=1}^{T} g_t(\mathbf{u}) \ .$$

Our main result is an algorithmic framework for the first player which guarantees low regret with respect to any vector $\mathbf{u} \in S$. Specifically, we derive regret bounds that take the following form

$$\forall \mathbf{u} \in S, \ \ \frac{1}{T}\sum_{t=1}^{T} g_t(\mathbf{w}_t) - \frac{1}{T}\sum_{t=1}^{T} g_t(\mathbf{u}) \ \leq \ \frac{f(\mathbf{u}) + L}{\sqrt{T}} \ , \tag{1}$$

where $f : S \to \mathbb{R}$ and $L \in \mathbb{R}_+$. Informally, the function $f$ measures the "complexity" of vectors in $S$ and the scalar $L$ is related to some generalized Lipschitz property of the functions $g_1, \ldots, g_T$. We defer the exact requirements we impose on $f$ and $L$ to later sections.

Our algorithmic framework emerges from a representation of the regret bound given in Eq. (1) using an optimization problem. Specifically, we rewrite Eq. (1) as follows

$$\frac{1}{T}\sum_{t=1}^{T} g_t(\mathbf{w}_t) \ \leq \ \inf_{\mathbf{u} \in S} \frac{1}{T}\sum_{t=1}^{T} g_t(\mathbf{u}) + \frac{f(\mathbf{u}) + L}{\sqrt{T}} \ . \tag{2}$$

That is, the average loss of the first player should be bounded above by the minimum value of an optimization problem in which we jointly minimize the average loss of $\mathbf{u}$ and the "complexity" of $\mathbf{u}$ as measured by the function $f$. Note that the optimization problem on the right-hand side of Eq. (2) can only be solved in hindsight after observing the entire sequence of loss functions. Nevertheless, writing the regret bound as in Eq. (2) implies that the average loss of the first player forms a lower bound for a minimization problem.

The notion of duality, commonly used in convex optimization theory, plays an important role in obtaining lower bounds for the minimal value of a minimization problem (see for example [14]). By generalizing the notion of Fenchel duality, we are able to derive a dual optimization problem, which can be optimized incrementally, as the game progresses. In order to derive explicit quantitative regret bounds we make an immediate use of the fact that dual objective lower bounds the primal objective. We therefore reduce the process of playing convex repeated games to the task of incrementally increasing the dual objective function. The amount by which the dual increases serves as a new and natural notion of progress. By doing so we are able to tie the primal objective value, the average loss of the first player, and the increase in the dual.

The rest of this paper is organized as follows. In Sec. 2 we establish our notation and point to a few mathematical tools that we use throughout the paper. Our main tool for deriving algorithms for playing convex repeated games is a generalization of Fenchel duality, described in Sec. 3. Our algorithmic framework is given in Sec. 4 and analyzed in Sec. 5. The generality of our framework allows us to utilize it in different problems arising in machine learning. Specifically, in Sec. 6 we underscore the applicability of our framework for online learning and in Sec. 7 we outline and analyze boosting algorithms based on our framework. We conclude with a discussion and point to related work in Sec. 8. Due to the lack of space, some of the details are omitted from the paper and can be found in [16].

## 2   Mathematical Background

We denote scalars with lower case letters (e.g. $x$ and $w$), and vectors with bold face letters (e.g. $\mathbf{x}$ and $\mathbf{w}$). The inner product between vectors $\mathbf{x}$ and $\mathbf{w}$ is denoted by $\langle \mathbf{x}, \mathbf{w} \rangle$. Sets are designated by upper case letters (e.g. $S$). The set of non-negative real numbers is denoted by $\mathbb{R}_+$. For any $k \geq 1$, the set of integers $\{1, \ldots, k\}$ is denoted by $[k]$. A norm of a vector $\mathbf{x}$ is denoted by $\|\mathbf{x}\|$. The dual norm is defined as $\|\boldsymbol{\lambda}\|_\star = \sup\{\langle \mathbf{x}, \boldsymbol{\lambda} \rangle : \|\mathbf{x}\| \leq 1\}$. For example, the Euclidean norm, $\|\mathbf{x}\|_2 = (\langle \mathbf{x}, \mathbf{x} \rangle)^{1/2}$ is dual to itself and the $\ell_1$ norm, $\|\mathbf{x}\|_1 = \sum_i |x_i|$, is dual to the $\ell_\infty$ norm, $\|\mathbf{x}\|_\infty = \max_i |x_i|$.

We next recall a few definitions from convex analysis. The reader familiar with convex analysis may proceed to Lemma 1 while for a more thorough introduction see for example [1]. A set $S$ is

convex if for any two vectors $\mathbf{w}_1, \mathbf{w}_2$ in $S$, all the line between $\mathbf{w}_1$ and $\mathbf{w}_2$ is also within $S$. That is, for any $\alpha \in [0, 1]$ we have that $\alpha \mathbf{w}_1 + (1 - \alpha)\mathbf{w}_2 \in S$. A set $S$ is open if every point in $S$ has a neighborhood lying in $S$. A set $S$ is closed if its complement is an open set. A function $f : S \to \mathbb{R}$ is closed and convex if for any scalar $\alpha \in \mathbb{R}$, the level set $\{\mathbf{w} : f(\mathbf{w}) \leq \alpha\}$ is closed and convex. The Fenchel conjugate of a function $f : S \to \mathbb{R}$ is defined as $f^\star(\boldsymbol{\theta}) = \sup_{\mathbf{w} \in S} \langle \mathbf{w}, \boldsymbol{\theta} \rangle - f(\mathbf{w})$ . If $f$ is closed and convex then the Fenchel conjugate of $f^\star$ is $f$ itself. The Fenchel-Young inequality states that for any $\mathbf{w}$ and $\boldsymbol{\theta}$ we have that $f(\mathbf{w}) + f^\star(\boldsymbol{\theta}) \geq \langle \mathbf{w}, \boldsymbol{\theta} \rangle$. A vector $\boldsymbol{\lambda}$ is a sub-gradient of a function $f$ at $\mathbf{w}$ if for all $\mathbf{w}' \in S$ we have that $f(\mathbf{w}') - f(\mathbf{w}) \geq \langle \mathbf{w}' - \mathbf{w}, \boldsymbol{\lambda} \rangle$. The differential set of $f$ at $\mathbf{w}$, denoted $\partial f(\mathbf{w})$, is the set of all sub-gradients of $f$ at $\mathbf{w}$. If $f$ is differentiable at $\mathbf{w}$ then $\partial f(\mathbf{w})$ consists of a single vector which amounts to the gradient of $f$ at $\mathbf{w}$ and is denoted by $\nabla f(\mathbf{w})$. Sub-gradients play an important role in the definition of Fenchel conjugate. In particular, the following lemma states that if $\boldsymbol{\lambda} \in \partial f(\mathbf{w})$ then Fenchel-Young inequality holds with equality.

**Lemma 1** *Let $f$ be a closed and convex function and let $\partial f(\mathbf{w}')$ be its differential set at $\mathbf{w}'$. Then, for all $\boldsymbol{\lambda}' \in \partial f(\mathbf{w}')$ we have, $f(\mathbf{w}') + f^\star(\boldsymbol{\lambda}') = \langle \boldsymbol{\lambda}', \mathbf{w}' \rangle$ .*

A continuous function $f$ is $\sigma$-strongly convex over a convex set $S$ with respect to a norm $\| \cdot \|$ if $S$ is contained in the domain of $f$ and for all $\mathbf{v}, \mathbf{u} \in S$ and $\alpha \in [0, 1]$ we have

$$f(\alpha \, \mathbf{v} + (1 - \alpha)\, \mathbf{u}) \leq \alpha \, f(\mathbf{v}) + (1 - \alpha) \, f(\mathbf{u}) - \frac{1}{2} \sigma \, \alpha \, (1 - \alpha) \, \|\mathbf{v} - \mathbf{u}\|^2 \ . \tag{3}$$

Strongly convex functions play an important role in our analysis primarily due to the following lemma.

**Lemma 2** *Let $\| \cdot \|$ be a norm over $\mathbb{R}^n$ and let $\| \cdot \|_\star$ be its dual norm. Let $f$ be a $\sigma$-strongly convex function on $S$ and let $f^\star$ be its Fenchel conjugate. Then, $f^\star$ is differentiable with $\nabla f^\star(\boldsymbol{\theta}) = \arg\max_{\mathbf{x} \in S} \langle \boldsymbol{\theta}, \mathbf{x} \rangle - f(\mathbf{x})$. Furthermore, for any $\boldsymbol{\theta}, \boldsymbol{\lambda} \in \mathbb{R}^n$ we have*

$$f^\star(\boldsymbol{\theta} + \boldsymbol{\lambda}) - f^\star(\boldsymbol{\theta}) \leq \langle \nabla f^\star(\boldsymbol{\theta}), \boldsymbol{\lambda} \rangle + \frac{1}{2\sigma} \|\boldsymbol{\lambda}\|_\star^2 \ .$$

Two notable examples of strongly convex functions which we use are as follows.

**Example 1** *The function $f(\mathbf{w}) = \frac{1}{2}\|\mathbf{w}\|_2^2$ is 1-strongly convex over $S = \mathbb{R}^n$ with respect to the $\ell_2$ norm. Its conjugate function is $f^\star(\boldsymbol{\theta}) = \frac{1}{2}\|\boldsymbol{\theta}\|_2^2$.*

**Example 2** *The function $f(\mathbf{w}) = \sum_{i=1}^n w_i \log(w_i / \frac{1}{n})$ is 1-strongly convex over the probabilistic simplex, $S = \{\mathbf{w} \in \mathbb{R}_+^n : \|\mathbf{w}\|_1 = 1\}$, with respect to the $\ell_1$ norm. Its conjugate function is $f^\star(\boldsymbol{\theta}) = \log(\frac{1}{n}\sum_{i=1}^n \exp(\theta_i))$.*

## 3 Generalized Fenchel Duality

In this section we derive our main analysis tool. We start by considering the following optimization problem,

$$\inf_{\mathbf{w} \in S} \ \left( c\, f(\mathbf{w}) + \sum_{t=1}^T g_t(\mathbf{w}) \right) \ ,$$

where $c$ is a non-negative scalar. An equivalent problem is

$$\inf_{\mathbf{w}_0, \mathbf{w}_1, \ldots, \mathbf{w}_T} \left( c\, f(\mathbf{w}_0) + \sum_{t=1}^T g_t(\mathbf{w}_t) \right) \ \text{s.t.} \ \mathbf{w}_0 \in S \ \text{and} \ \forall t \in [T], \ \mathbf{w}_t = \mathbf{w}_0 \ .$$

Introducing $T$ vectors $\boldsymbol{\lambda}_1, \ldots, \boldsymbol{\lambda}_T$, each $\boldsymbol{\lambda}_t \in \mathbb{R}^n$ is a vector of Lagrange multipliers for the equality constraint $\mathbf{w}_t = \mathbf{w}_0$, we obtain the following Lagrangian

$$\mathcal{L}(\mathbf{w}_0, \mathbf{w}_1, \ldots, \mathbf{w}_T, \boldsymbol{\lambda}_1, \ldots, \boldsymbol{\lambda}_T) = c\, f(\mathbf{w}_0) + \sum_{t=1}^T g_t(\mathbf{w}_t) + \sum_{t=1}^T \langle \boldsymbol{\lambda}_t, \mathbf{w}_0 - \mathbf{w}_t \rangle \ .$$

The dual problem is the task of maximizing the following dual objective value,

$$
\begin{aligned}
\mathcal{D}(\boldsymbol{\lambda}_1, \ldots, \boldsymbol{\lambda}_T) &= \inf_{\mathbf{w}_0 \in S, \mathbf{w}_1, \ldots, \mathbf{w}_T} \mathcal{L}(\mathbf{w}_0, \mathbf{w}_1, \ldots, \mathbf{w}_T, \boldsymbol{\lambda}_1, \ldots, \boldsymbol{\lambda}_T) \\
&= -c \sup_{\mathbf{w}_0 \in S} \left( \langle \mathbf{w}_0, -\tfrac{1}{c}\sum_{t=1}^T \boldsymbol{\lambda}_t \rangle - f(\mathbf{w}_0) \right) - \sum_{t=1}^T \sup_{\mathbf{w}_t} \left( \langle \mathbf{w}_t, \boldsymbol{\lambda}_t \rangle - g_t(\mathbf{w}_t) \right) \\
&= -c\, f^\star\left( -\tfrac{1}{c}\sum_{t=1}^T \boldsymbol{\lambda}_t \right) - \sum_{t=1}^T g_t^\star(\boldsymbol{\lambda}_t) \ ,
\end{aligned}
$$

where, following the exposition of Sec. 2, $f^\star, g_1^\star, \ldots, g_T^\star$ are the Fenchel conjugate functions of $f, g_1, \ldots, g_T$. Therefore, the generalized Fenchel dual problem is

$$\sup_{\boldsymbol{\lambda}_1, \ldots, \boldsymbol{\lambda}_T} - c\, f^\star \left( -\tfrac{1}{c} \sum_{t=1}^T \boldsymbol{\lambda}_t \right) - \sum_{t=1}^T g_t^\star(\boldsymbol{\lambda}_t) \quad . \tag{4}$$

Note that when $T = 1$ and $c = 1$, the above duality is the so called Fenchel duality.

## 4 A Template Learning Algorithm for Convex Repeated Games

In this section we describe a template learning algorithm for playing convex repeated games. As mentioned before, we study convex repeated games from the viewpoint of the first player which we shortly denote as P1. Recall that we would like our learning algorithm to achieve a regret bound of the form given in Eq. (2). We start by rewriting Eq. (2) as follows

$$\sum_{t=1}^T g_t(\mathbf{w}_t) - c\, L \;\leq\; \inf_{\mathbf{u} \in S} \left( c\, f(\mathbf{u}) + \sum_{t=1}^m g_t(\mathbf{u}) \right) \quad , \tag{5}$$

where $c = \sqrt{T}$. Thus, up to the sublinear term $c\, L$, the cumulative loss of P1 lower bounds the optimum of the minimization problem on the right-hand side of Eq. (5). In the previous section we derived the generalized Fenchel dual of the right-hand side of Eq. (5). Our construction is based on the weak duality theorem stating that any value of the dual problem is smaller than the optimum value of the primal problem. The algorithmic framework we propose is therefore derived by incrementally ascending the dual objective function. Intuitively, by ascending the dual objective we move closer to the optimal primal value and therefore our performance becomes similar to the performance of the best fixed weight vector which minimizes the right-hand side of Eq. (5).

Initially, we use the elementary dual solution $\boldsymbol{\lambda}_t^1 = \mathbf{0}$ for all $t$. We assume that $\inf_{\mathbf{w}} f(\mathbf{w}) = 0$ and for all $t$ $\inf_{\mathbf{w}} g_t(\mathbf{w}) = 0$ which imply that $\mathcal{D}(\boldsymbol{\lambda}_1^1, \ldots, \boldsymbol{\lambda}_T^1) = 0$. We assume in addition that $f$ is $\sigma$-strongly convex. Therefore, based on Lemma 2, the function $f^\star$ is differentiable. At trial $t$, P1 uses for prediction the vector

$$\mathbf{w}_t \;=\; \nabla f^\star \left( -\tfrac{1}{c} \sum_{i=1}^T \boldsymbol{\lambda}_i^t \right) \quad . \tag{6}$$

After predicting $\mathbf{w}_t$, P1 receives the function $g_t$ and suffers the loss $g_t(\mathbf{w}_t)$. Then, P1 updates the dual variables as follows. Denote by $\partial_t$ the differential set of $g_t$ at $\mathbf{w}_t$, that is,

$$\partial_t \;=\; \{\boldsymbol{\lambda} \,:\, \forall \mathbf{w} \in S,\; g_t(\mathbf{w}) - g_t(\mathbf{w}_t) \;\geq\; \langle \boldsymbol{\lambda}, \mathbf{w} - \mathbf{w}_t \rangle \} \quad . \tag{7}$$

The new dual variables $(\boldsymbol{\lambda}_1^{t+1}, \ldots, \boldsymbol{\lambda}_T^{t+1})$ are set to be any set of vectors which satisfy the following two conditions:

$$\begin{aligned} &(i). \quad \exists \boldsymbol{\lambda}' \in \partial_t \;\; \text{s.t.} \;\; \mathcal{D}(\boldsymbol{\lambda}_1^{t+1}, \ldots, \boldsymbol{\lambda}_T^{t+1}) \;\geq\; \mathcal{D}(\boldsymbol{\lambda}_1^t, \ldots, \boldsymbol{\lambda}_{t-1}^t, \boldsymbol{\lambda}', \boldsymbol{\lambda}_{t+1}^t, \ldots, \boldsymbol{\lambda}_T^t) \\ &(ii). \quad \forall i > t,\; \boldsymbol{\lambda}_i^{t+1} = \mathbf{0} \end{aligned} \quad . \tag{8}$$

In the next section we show that condition (i) ensures that the increase of the dual at trial $t$ is proportional to the loss $g_t(\mathbf{w}_t)$. The second condition ensures that we can actually calculate the dual at trial $t$ without any knowledge on the yet to be seen loss functions $g_{t+1}, \ldots, g_T$.

We conclude this section with two update rules that trivially satisfy the above two conditions. The first update scheme simply finds $\boldsymbol{\lambda}' \in \partial_t$ and set

$$\boldsymbol{\lambda}_i^{t+1} \;=\; \begin{cases} \boldsymbol{\lambda}' & \text{if } i = t \\ \boldsymbol{\lambda}_i^t & \text{if } i \neq t \end{cases} \quad . \tag{9}$$

The second update defines

$$(\boldsymbol{\lambda}_1^{t+1}, \ldots, \boldsymbol{\lambda}_T^{t+1}) \;=\; \operatorname*{argmax}_{\boldsymbol{\lambda}_1, \ldots, \boldsymbol{\lambda}_T} \mathcal{D}(\boldsymbol{\lambda}_1, \ldots, \boldsymbol{\lambda}_T) \quad \text{s.t.} \quad \forall i \neq t,\; \boldsymbol{\lambda}_i = \boldsymbol{\lambda}_i^t \quad . \tag{10}$$

# 5 Analysis

In this section we analyze the performance of the template algorithm given in the previous section. Our proof technique is based on monitoring the value of the dual objective function. The main result is the following lemma which gives upper and lower bounds for the final value of the dual objective function.

**Lemma 3** *Let $f$ be a $\sigma$-strongly convex function with respect to a norm $\|\cdot\|$ over a set $S$ and assume that $\min_{\mathbf{w}\in S} f(\mathbf{w}) = 0$. Let $g_1,\ldots,g_T$ be a sequence of convex and closed functions such that $\inf_{\mathbf{w}} g_t(\mathbf{w}) = 0$ for all $t \in [T]$. Suppose that a dual-incrementing algorithm which satisfies the conditions of Eq. (8) is run with $f$ as a complexity function on the sequence $g_1,\ldots,g_T$. Let $\mathbf{w}_1,\ldots,\mathbf{w}_T$ be the sequence of primal vectors that the algorithm generates and $\boldsymbol{\lambda}_1^{T+1},\ldots,\boldsymbol{\lambda}_T^{T+1}$ be its final sequence of dual variables. Then, there exists a sequence of sub-gradients $\boldsymbol{\lambda}_1',\ldots,\boldsymbol{\lambda}_T'$, where $\boldsymbol{\lambda}_t' \in \partial_t$ for all $t$, such that*

$$\sum_{t=1}^T g_t(\mathbf{w}_t) - \frac{1}{2\,\sigma\,c}\sum_{t=1}^T \|\boldsymbol{\lambda}_t'\|_\star^2 \;\leq\; \mathcal{D}(\boldsymbol{\lambda}_1^{T+1},\ldots,\boldsymbol{\lambda}_T^{T+1}) \;\leq\; \inf_{\mathbf{w}\in S} c\,f(\mathbf{w}) + \sum_{t=1}^T g_t(\mathbf{w}) \;.$$

**Proof** The second inequality follows directly from the weak duality theorem. Turning to the left most inequality, denote $\Delta_t = \mathcal{D}(\boldsymbol{\lambda}_1^{t+1},\ldots,\boldsymbol{\lambda}_T^{t+1}) - \mathcal{D}(\boldsymbol{\lambda}_1^t,\ldots,\boldsymbol{\lambda}_T^t)$ and note that $\mathcal{D}(\boldsymbol{\lambda}_1^{T+1},\ldots,\boldsymbol{\lambda}_T^{T+1})$ can be rewritten as

$$\mathcal{D}(\boldsymbol{\lambda}_1^{T+1},\ldots,\boldsymbol{\lambda}_T^{T+1}) \;=\; \textstyle\sum_{t=1}^T \Delta_t \,-\, \mathcal{D}(\boldsymbol{\lambda}_1^1,\ldots,\boldsymbol{\lambda}_T^1) \;=\; \textstyle\sum_{t=1}^T \Delta_t \;, \tag{11}$$

where the last equality follows from the fact that $f^\star(\mathbf{0}) = g_1^\star(\mathbf{0}) = \ldots = g_T^\star(\mathbf{0}) = 0$. The definition of the update implies that $\Delta_t \geq \mathcal{D}(\boldsymbol{\lambda}_1^t,\ldots,\boldsymbol{\lambda}_{t-1}^t,\boldsymbol{\lambda}_t',\mathbf{0},\ldots,\mathbf{0}) - \mathcal{D}(\boldsymbol{\lambda}_1^t,\ldots,\boldsymbol{\lambda}_{t-1}^t,\mathbf{0},\mathbf{0},\ldots,\mathbf{0})$ for some subgradient $\boldsymbol{\lambda}_t' \in \partial_t$. Denoting $\boldsymbol{\theta}_t = -\frac{1}{c}\sum_{j=1}^{t-1}\boldsymbol{\lambda}_j$, we now rewrite the lower bound on $\Delta_t$ as, $\Delta_t \geq -c\,(f^\star(\boldsymbol{\theta}_t - \boldsymbol{\lambda}_t'/c) - f^\star(\boldsymbol{\theta}_t)) - g_t^\star(\boldsymbol{\lambda}_t')$. Using Lemma 2 and the definition of $\mathbf{w}_t$ we get that

$$\Delta_t \;\geq\; \langle \mathbf{w}_t, \boldsymbol{\lambda}_t'\rangle - g_t^\star(\boldsymbol{\lambda}_t') - \frac{1}{2\,\sigma\,c}\|\boldsymbol{\lambda}_t'\|_\star^2 \;. \tag{12}$$

Since $\boldsymbol{\lambda}_t' \in \partial_t$ and since we assume that $g_t$ is closed and convex, we can apply Lemma 1 to get that $\langle \mathbf{w}_t, \boldsymbol{\lambda}_t'\rangle - g_t^\star(\boldsymbol{\lambda}_t') = g_t(\mathbf{w}_t)$. Plugging this equality into Eq. (12) and summing over $t$ we obtain that

$$\textstyle\sum_{t=1}^T \Delta_t \;\geq\; \textstyle\sum_{t=1}^T g_t(\mathbf{w}_t) - \frac{1}{2\,\sigma\,c}\sum_{t=1}^T \|\boldsymbol{\lambda}_t'\|_\star^2 \;.$$

Combining the above inequality with Eq. (11) concludes our proof. ∎

The following regret bound follows as a direct corollary of Lemma 3.

**Theorem 1** *Under the same conditions of Lemma 3. Denote $L = \frac{1}{T}\sum_{t=1}^T \|\boldsymbol{\lambda}_t'\|_\star^2$. Then, for all $\mathbf{w}\in S$ we have,*

$$\tfrac{1}{T}\textstyle\sum_{t=1}^T g_t(\mathbf{w}_t) - \tfrac{1}{T}\textstyle\sum_{t=1}^T g_t(\mathbf{w}) \;\leq\; \frac{c\,f(\mathbf{w})}{T} + \frac{L}{2\,\sigma\,c} \;.$$

*In particular, if $c = \sqrt{T}$, we obtain the bound,*

$$\tfrac{1}{T}\textstyle\sum_{t=1}^T g_t(\mathbf{w}_t) - \tfrac{1}{T}\textstyle\sum_{t=1}^T g_t(\mathbf{w}) \;\leq\; \frac{f(\mathbf{w})+L/(2\,\sigma)}{\sqrt{T}} \;.$$

# 6 Application to Online learning

In Sec. 1 we cast the task of online learning as a convex repeated game. We now demonstrate the applicability of our algorithmic framework for the problem of instance ranking. We analyze this setting since several prediction problems, including binary classification, multiclass prediction, multilabel prediction, and label ranking, can be cast as special cases of the instance ranking problem.

Recall that on each online round, the learner receives a question-answer pair. In instance ranking, the question is encoded by a matrix $X_t$ of dimension $k_t \times n$ and the answer is a vector $\mathbf{y}_t \in \mathbb{R}^{k_t}$. The semantic of $\mathbf{y}_t$ is as follows. For any pair $(i,j)$, if $y_{t,i} > y_{t,j}$ then we say that $\mathbf{y}_t$ *ranks*

the $i$'th row of $X_t$ ahead of the $j$'th row of $X_t$. We also interpret $y_{t,i} - y_{t,j}$ as the confidence in which the $i$'th row should be ranked ahead of the $j$'th row. For example, each row of $X_t$ encompasses a representation of a movie while $y_{t,i}$ is the movie's rating, expressed as the number of stars this movie has received by a movie reviewer. The predictions of the learner are determined based on a weight vector $\mathbf{w}_t \in \mathbb{R}^n$ and are defined to be $\hat{\mathbf{y}}_t = X_t \mathbf{w}_t$. Finally, let us define two loss functions for ranking, both generalize the hinge-loss used in binary classification problems. Denote by $E_t$ the set $\{(i,j) : y_{t,i} > y_{t,j}\}$. For all $(i,j) \in E_t$ we define a pair-based hinge-loss $\ell_{i,j}(\mathbf{w};(X_t,\mathbf{y}_t)) = [(y_{t,i} - y_{t,j}) - \langle \mathbf{w}, \mathbf{x}_{t,i} - \mathbf{x}_{t,j} \rangle]_+$, where $[a]_+ = \max\{a,0\}$ and $\mathbf{x}_{t,i}, \mathbf{x}_{t,j}$ are respectively the $i$'th and $j$'th rows of $X_t$. Note that $\ell_{i,j}$ is zero if $\mathbf{w}$ ranks $\mathbf{x}_{t,i}$ higher than $\mathbf{x}_{t,j}$ with a sufficient confidence. Ideally, we would like $\ell_{i,j}(\mathbf{w}_t;(X_t,\mathbf{y}_t))$ to be zero for all $(i,j) \in E_t$. If this is not the case, we are being penalized according to some combination of the pair-based losses $\ell_{i,j}$. For example, we can set $\ell(\mathbf{w};(X_t,\mathbf{y}_t))$ to be the average over the pair losses, $\ell^{\text{avg}}(\mathbf{w};(X_t,\mathbf{y}_t)) = \frac{1}{|E_t|}\sum_{(i,j)\in E_t}\ell_{i,j}(\mathbf{w};(X_t,\mathbf{y}_t))$ . This loss was suggested by several authors (see for example [18]). Another popular approach (see for example [5]) penalizes according to the maximal loss over the individual pairs, $\ell^{\text{max}}(\mathbf{w};(X_t,\mathbf{y}_t)) = \max_{(i,j)\in E_t}\ell_{i,j}(\mathbf{w};(X_t,\mathbf{y}_t))$ . We can apply our algorithmic framework given in Sec. 4 for ranking, using for $g_t(\mathbf{w})$ either $\ell^{\text{avg}}(\mathbf{w};(X_t,\mathbf{y}_t))$ or $\ell^{\text{max}}(\mathbf{w};(X_t,\mathbf{y}_t))$. The following theorem provides us with a sufficient condition under which the regret bound from Thm. 1 holds for ranking as well.

**Theorem 2** *Let $f$ be a $\sigma$-strongly convex function over $S$ with respect to a norm $\|\cdot\|$. Denote by $L_t$ the maximum over $(i,j) \in E_t$ of $\|\mathbf{x}_{t,i} - \mathbf{x}_{t,j}\|_*^2$. Then, for both $g_t(\mathbf{w}) = \ell^{\text{avg}}(\mathbf{w};(X_t,\mathbf{y}_t))$ and $g_t(\mathbf{w}) = \ell^{\text{max}}(\mathbf{w};(X_t,\mathbf{y}_t))$, the following regret bound holds*

$$\forall \mathbf{u} \in S, \ \ \frac{1}{T}\sum_{t=1}^{T}g_t(\mathbf{w}_t) - \frac{1}{T}\sum_{t=1}^{T}g_t(\mathbf{u}) \ \leq \ \frac{f(\mathbf{u}) + \frac{1}{T}\sum_{t=1}^{T}L_t/(2\,\sigma)}{\sqrt{T}} \ \ .$$

# 7 The Boosting Game

In this section we describe the applicability of our algorithmic framework to the analysis of boosting algorithms. A boosting algorithm uses a weak learning algorithm that generates weak-hypotheses whose performances are just slightly better than random guessing to build a strong-hypothesis which can attain an arbitrarily low error. The AdaBoost algorithm, proposed by Freund and Schapire [6], receives as input a training set of examples $\{(\mathbf{x}_1,y_1),\ldots,(\mathbf{x}_m,y_m)\}$ where for all $i \in [m]$, $\mathbf{x}_i$ is taken from an instance domain $\mathcal{X}$, and $y_i$ is a binary label, $y_i \in \{+1,-1\}$. The boosting process proceeds in a sequence of consecutive trials. At trial $t$, the booster first defines a distribution, denoted $\mathbf{w}_t$, over the set of examples. Then, the booster passes the training set along with the distribution $\mathbf{w}_t$ to the weak learner. The weak learner is assumed to return a hypothesis $h_t : \mathcal{X} \to \{+1,-1\}$ whose average error is slightly smaller than $\frac{1}{2}$. That is, there exists a constant $\gamma > 0$ such that,

$$\epsilon_t \stackrel{def}{=} \sum_{i=1}^{m} w_{t,i}\frac{1 - y_i h_t(\mathbf{x}_i)}{2} \ \leq \ \frac{1}{2} - \gamma \ . \tag{13}$$

The goal of the boosting algorithm is to invoke the weak learner several times with different distributions, and to combine the hypotheses returned by the weak learner into a final, so called strong, hypothesis whose error is small. The final hypothesis combines linearly the $T$ hypotheses returned by the weak learner with coefficients $\alpha_1,\ldots,\alpha_T$, and is defined to be the sign of $h_f(\mathbf{x})$ where $h_f(\mathbf{x}) = \sum_{t=1}^{T}\alpha_t h_t(\mathbf{x})$ . The coefficients $\alpha_1,\ldots,\alpha_T$ are determined by the booster. In AdaBoost, the initial distribution is set to be the uniform distribution, $\mathbf{w}_1 = (\frac{1}{m},\ldots,\frac{1}{m})$. At iteration $t$, the value of $\alpha_t$ is set to be $\frac{1}{2}\log((1 - \epsilon_t)/\epsilon_t)$. The distribution is updated by the rule $w_{t+1,i} = w_{t,i}\exp(-\alpha_t y_i h_t(\mathbf{x}_i))/Z_t$, where $Z_t$ is a normalization factor. Freund and Schapire [6] have shown that under the assumption given in Eq. (13), the error of the final strong hypothesis is at most $\exp(-2\gamma^2 T)$.

Several authors [15, 13, 8, 4] have proposed to view boosting as a coordinate-wise greedy optimization process. To do so, note first that $h_f$ errs on an example $(\mathbf{x},y)$ iff $y\,h_f(\mathbf{x}) \leq 0$. Therefore, the exp-loss function, defined as $\exp(-y\,h_f(\mathbf{x}))$, is a smooth upper bound of the zero-one error, which equals to 1 if $y\,h_f(\mathbf{x}) \leq 0$ and to 0 otherwise. Thus, we can restate the goal of boosting as minimizing the average exp-loss of $h_f$ over the training set with respect to the variables $\alpha_1,\ldots,\alpha_T$. To simplify our derivation in the sequel, we prefer to say that boosting maximizes the negation of the loss, that is,

$$\max_{\alpha_1,\ldots,\alpha_T} \ -\frac{1}{m}\sum_{i=1}^{m}\exp\left(-y_i\sum_{t=1}^{T}\alpha_t h_t(\mathbf{x}_i)\right) \ . \tag{14}$$

In this view, boosting is an optimization procedure which iteratively maximizes Eq. (14) with respect to the variables $\alpha_1, \ldots, \alpha_T$. This view of boosting, enables the hypotheses returned by the weak learner to be general functions into the reals, $h_t : \mathcal{X} \to \mathbb{R}$ (see for instance [15]).

In this paper we view boosting as a convex repeated game between a booster and a weak learner. To motivate our construction, we would like to note that boosting algorithms define weights in two different domains: the vectors $\mathbf{w}_t \in \mathbb{R}^m$ which assign weights to *examples* and the weights $\{\alpha_t : t \in [T]\}$ over *weak-hypotheses*. In the terminology used throughout this paper, the weights $\mathbf{w}_t \in \mathbb{R}^m$ are *primal* vectors while (as we show in the sequel) each weight $\alpha_t$ of the hypothesis $h_t$ is related to a *dual* vector $\boldsymbol{\lambda}_t$. In particular, we show that Eq. (14) is exactly the Fenchel dual of a primal problem for a convex repeated game, thus the algorithmic framework described thus far for playing games naturally fits the problem of iteratively solving Eq. (14).

To derive the primal problem whose Fenchel dual is the problem given in Eq. (14) let us first denote by $\mathbf{v}_t$ the vector in $\mathbb{R}^m$ whose $i$th element is $v_{t,i} = y_i h_t(\mathbf{x}_i)$. For all $t$, we set $g_t$ to be the function $g_t(\mathbf{w}) = [\langle \mathbf{w}, \mathbf{v}_t \rangle]_+$. Intuitively, $g_t$ penalizes vectors $\mathbf{w}$ which assign large weights to examples which are predicted accurately, that is $y_i h_t(\mathbf{x}_i) > 0$. In particular, if $h_t(\mathbf{x}_i) \in \{+1, -1\}$ and $\mathbf{w}_t$ is a distribution over the $m$ examples (as is the case in AdaBoost), $g_t(\mathbf{w}_t)$ reduces to $1 - 2\epsilon_t$ (see Eq. (13)). In this case, minimizing $g_t$ is equivalent to maximizing the error of the individual hypothesis $h_t$ over the examples. Consider the problem of minimizing $c f(\mathbf{w}) + \sum_{t=1}^T g_t(\mathbf{w})$ where $f(\mathbf{w})$ is the relative entropy given in Example 2 and $c = 1/(2\gamma)$ (see Eq. (13)). To derive its Fenchel dual, we note that $g_t^\star(\boldsymbol{\lambda}_t) = 0$ if there exists $\beta_t \in [0, 1]$ such that $\boldsymbol{\lambda}_t = \beta_t \mathbf{v}_t$ and otherwise $g_t^\star(\boldsymbol{\lambda}_t) = \infty$ (see [16]). In addition, let us define $\alpha_t = 2\gamma\beta_t$. Since our goal is to maximize the dual, we can restrict $\boldsymbol{\lambda}_t$ to take the form $\boldsymbol{\lambda}_t = \beta_t \mathbf{v}_t = \frac{\alpha_t}{2\gamma} \mathbf{v}_t$, and get that

$$\mathcal{D}(\boldsymbol{\lambda}_1, \ldots, \boldsymbol{\lambda}_T) = -c f^\star \left( -\frac{1}{c} \sum_{t=1}^T \beta_t \mathbf{v}_t \right) = -\frac{1}{2\gamma} \log \left( \frac{1}{m} \sum_{i=1}^m e^{-\sum_{t=1}^T \alpha_t y_i h_t(\mathbf{x}_i)} \right) . \quad (15)$$

Minimizing the exp-loss of the strong hypothesis is therefore the dual problem of the following primal minimization problem: find a distribution over the examples, whose relative entropy to the uniform distribution is as small as possible while the correlation of the distribution with each $\mathbf{v}_t$ is as small as possible. Since the correlation of $\mathbf{w}$ with $\mathbf{v}_t$ is inversely proportional to the error of $h_t$ with respect to $\mathbf{w}$, we obtain that in the primal problem we are trying to *maximize* the error of each *individual* hypothesis, while in the dual problem we *minimize* the global error of the *strong* hypothesis. The intuition of finding distributions which in retrospect result in large error rates of individual hypotheses was also alluded in [15, 8].

We can now apply our algorithmic framework from Sec. 4 to boosting. We describe the game with the parameters $\alpha_t$, where $\alpha_t \in [0, 2\gamma]$, and underscore that in our case, $\boldsymbol{\lambda}_t = \frac{\alpha_t}{2\gamma} \mathbf{v}_t$. At the beginning of the game the booster sets all dual variables to be zero, $\forall t \ \alpha_t = 0$. At trial $t$ of the boosting game, the booster first constructs a primal weight vector $\mathbf{w}_t \in \mathbb{R}^m$, which assigns importance weights to the examples in the training set. The primal vector $\mathbf{w}_t$ is constructed as in Eq. (6), that is, $\mathbf{w}_t = \nabla f^\star(\boldsymbol{\theta}_t)$, where $\boldsymbol{\theta}_t = -\sum_i \alpha_i \mathbf{v}_i$. Then, the weak learner responds by presenting the loss function $g_t(\mathbf{w}) = [\langle \mathbf{w}, \mathbf{v}_t \rangle]_+$. Finally, the booster updates the dual variables so as to increase the dual objective function. It is possible to show that if the range of $h_t$ is $\{+1, -1\}$ then the update given in Eq. (10) is equivalent to the update $\alpha_t = \min\{2\gamma, \frac{1}{2}\log((1 - \epsilon_t)/\epsilon_t)\}$. We have thus obtained a variant of AdaBoost in which the weights $\alpha_t$ are capped above by $2\gamma$. A disadvantage of this variant is that we need to know the parameter $\gamma$. We would like to note in passing that this limitation can be lifted by a different definition of the functions $g_t$. We omit the details due to the lack of space.

To analyze our game of boosting, we note that the conditions given in Lemma 3 holds and therefore the left-hand side inequality given in Lemma 3 tells us that $\sum_{t=1}^T g_t(\mathbf{w}_t) - \frac{1}{2c} \sum_{t=1}^T \|\boldsymbol{\lambda}_t'\|_\infty^2 \leq \mathcal{D}(\boldsymbol{\lambda}_1^{T+1}, \ldots, \boldsymbol{\lambda}_T^{T+1})$. The definition of $g_t$ and the weak learnability assumption given in Eq. (13) imply that $\langle \mathbf{w}_t, \mathbf{v}_t \rangle \geq 2\gamma$ for all $t$. Thus, $g_t(\mathbf{w}_t) = \langle \mathbf{w}_t, \mathbf{v}_t \rangle \geq 2\gamma$ which also implies that $\boldsymbol{\lambda}_t' = \mathbf{v}_t$. Recall that $v_{t,i} = y_i h_t(\mathbf{x}_i)$. Assuming that the range of $h_t$ is $[+1, -1]$ we get that $\|\boldsymbol{\lambda}_t'\|_\infty \leq 1$. Combining all the above with the left-hand side inequality given in Lemma 3 we get that $2T\gamma - \frac{T}{2c} \leq \mathcal{D}(\boldsymbol{\lambda}_1^{T+1}, \ldots, \boldsymbol{\lambda}_T^{T+1})$. Using the definition of $\mathcal{D}$ (see Eq. (15)), the value $c = 1/(2\gamma)$, and rearranging terms we recover the original bound for AdaBoost $\frac{1}{m} \sum_{i=1}^m e^{-y_i \sum_{t=1}^T \alpha_t h_t(\mathbf{x}_i)} \leq e^{-2\gamma^2 T}$ .

# 8 Related Work and Discussion

We presented a new framework for designing and analyzing algorithms for playing convex repeated games. Our framework was used for the analysis of known algorithms for both online learning and boosting settings. The framework also paves the way to new algorithms. In a previous paper [17], we suggested the use of duality for the design of online algorithms in the context of mistake bound analysis. The contribution of this paper over [17] is three fold as we now briefly discuss.

First, we generalize the applicability of the framework beyond the specific setting of online learning with the hinge-loss to the general setting of convex repeated games. The setting of convex repeated games was formally termed "online convex programming" by Zinkevich [19] and was first presented by Gordon in [9]. There is voluminous amount of work on unifying approaches for deriving online learning algorithms. We refer the reader to [11, 12, 3] for work closely related to the content of this paper. By generalizing our previously studied algorithmic framework [17] beyond online learning, we can automatically utilize well known online learning algorithms, such as the EG and p-norm algorithms [12, 11], to the setting of online convex programming. We would like to note that the algorithms presented in [19] can be derived as special cases of our algorithmic framework by setting $f(\mathbf{w}) = \frac{1}{2}\|\mathbf{w}\|^2$. Parallel and independently to this work, Gordon [10] described another algorithmic framework for online convex programming that is closely related to the potential based algorithms described by Cesa-Bianchi and Lugosi [3]. Gordon also considered the problem of defining appropriate potential functions. Our work generalizes some of the theorems in [10] while providing a somewhat simpler analysis.

Second, the usage of generalized Fenchel duality rather than the Lagrange duality given in [17] enables us to analyze boosting algorithms based on the framework. Many authors derived unifying frameworks for boosting algorithms [13, 8, 4]. Nonetheless, our general framework and the connection between game playing and Fenchel duality underscores an interesting perspective of both online learning and boosting. We believe that this viewpoint has the potential of yielding new algorithms in both domains.

Last, despite the generality of the framework introduced in this paper, the resulting analysis is more distilled than the earlier analysis given in [17] for two reasons. (i) The usage of Lagrange duality in [17] is somehow restricted while the notion of generalized Fenchel duality is more appropriate to the general and broader problems we consider in this paper. (ii) The strongly convex property we employ both simplifies the analysis and enables more intuitive conditions in our theorems.

There are various possible extensions of the work that we did not pursue here due to the lack of space. For instanc, our framework can naturally be used for the analysis of other settings such as repeated games (see [7, 19]). The applicability of our framework to online learning can also be extended to other prediction problems such as regression and sequence prediction. Last, we conjecture that our primal-dual view of boosting will lead to new methods for regularizing boosting algorithms, thus improving their generalization capabilities.

## References

[1] J. Borwein and A. Lewis. *Convex Analysis and Nonlinear Optimization*. Springer, 2006.

[2] S. Boyd and L. Vandenberghe. *Convex Optimization*. Cambridge University Press, 2004.

[3] N. Cesa-Bianchi and G. Lugosi. *Prediction, learning, and games*. Cambridge University Press, 2006.

[4] M. Collins, R.E. Schapire, and Y. Singer. Logistic regression, AdaBoost and Bregman distances. *Machine Learning*, 2002.

[5] K. Crammer, O. Dekel, J. Keshet, S. Shalev-Shwartz, and Y. Singer. Online passive aggressive algorithms. *JMLR*, 7, Mar 2006.

[6] Y. Freund and R.E. Schapire. A decision-theoretic generalization of on-line learning and an application to boosting. In *EuroCOLT*, 1995.

[7] Y. Freund and R.E. Schapire. Game theory, on-line prediction and boosting. In *COLT*, 1996.

[8] J. Friedman, T. Hastie, and R. Tibshirani. Additive logistic regression: a statistical view of boosting. *Annals of Statistics*, 28(2), 2000.

[9] G. Gordon. Regret bounds for prediction problems. In *COLT*, 1999.

[10] G. Gordon. No-regret algorithms for online convex programs. In *NIPS*, 2006.

[11] A. J. Grove, N. Littlestone, and D. Schuurmans. General convergence results for linear discriminant updates. *Machine Learning*, 43(3), 2001.

[12] J. Kivinen and M. Warmuth. Relative loss bounds for multidimensional regression problems. *Journal of Machine Learning*, 45(3),2001.

[13] L. Mason, J. Baxter, P. Bartlett, and M. Frean. Functional gradient techniques for combining hypotheses. In *Advances in Large Margin Classifiers*. MIT Press, 1999.

[14] Y. Nesterov. Primal-dual subgradient methods for convex problems. Technical report, Center for Operations Research and Econometrics (CORE), Catholic University of Louvain (UCL), 2005.

[15] R. E. Schapire and Y. Singer. Improved boosting algorithms using confidence-rated predictions. *Machine Learning*, 37(3):1–40, 1999.

[16] S. Shalev-Shwartz and Y. Singer. Convex repeated games and fenchel duality. Technical report, The Hebrew University, 2006.

[17] S. Shalev-Shwartz and Y. Singer. Online learning meets optimization in the dual. In *COLT*, 2006.

[18] J. Weston and C. Watkins. Support vector machines for multi-class pattern recognition. In *ESANN*, April 1999.

[19] M. Zinkevich. Online convex programming and generalized infinitesimal gradient ascent. In *ICML*, 2003.
